# Bifurcation Analysis of a Silicon Neuron

**Girish N. Patel[1], Gennady S. Cymbalyuk[2,3],**
**Ronald L. Calabrese[2], and Stephen P. DeWeerth[1]**

[1]School of Electrical and Computer Engineering
Georgia Institute of Technology
Atlanta, Ga. 30332-0250
{girish.patel, steve.deweerth}@ece.gatech.edu

[2]Department of Biology
Emory University
1510 Clifton Road, Atlanta, GA 30322
{gcym, rcalabre}@biology.emory.edu

[3]Institute of Mathematical Problems in Biology RAS
Pushchino, Moscow Region, Russia 142292 (on leave)

## Abstract

We have developed a VLSI silicon neuron and a corresponding mathematical model that is a two state-variable system. We describe the circuit implementation and compare the behaviors observed in the silicon neuron and the mathematical model. We also perform bifurcation analysis of the mathematical model by varying the externally applied current and show that the behaviors exhibited by the silicon neuron under corresponding conditions are in good agreement to those predicted by the bifurcation analysis.

## 1 Introduction

The use of hardware models to understand dynamical behaviors in biological systems is an approach that has a long and fruitful history [1][2]. The implementation in silicon of oscillatory neural networks that model rhythmic motor-pattern generation in animals is one recent addition to these modeling efforts [3][4]. The oscillatory patterns generated by these systems result from intrinsic membrane properties of individual neurons and their synaptic interactions within the network [5]. As the complexity of these oscillatory silicon systems increases, effective mathematical analysis becomes increasingly more important to our understanding their behavior. However, the nonlinear dynamical behaviors of the model neurons and the large-scale interconnectivity among these neurons makes it very difficult to analyze theoretically the behavior of the resulting very large-scale integrated (VLSI) systems. Thus, it is important to first identify methods for modeling the model neurons that underlie these oscillatory systems.

Several simplified neuronal models have been used in the mathematical simulations of pattern generating networks [6][7][8]. In this paper, we describe the implementation of a

two-state-variable silicon neuron that has been used effectively to develop oscillatory networks [9][10]. We then derive a mathematical model of this implementation and analyze the neuron and the model using nonlinear dynamical techniques including bifurcation analysis [11]. Finally, we compare the experimental data derived from the silicon neuron to that obtained from the mathematical model.

## 2   The silicon model neuron

The schematic for our silicon model neuron is shown in Figure 1. This silicon neuron is inspired by the two-state, Morris–Lecar neuron model [12][13]. Transistor $M_1$, analogous to the voltage-gated calcium channel in the Morris–Lecar model, provides an instantaneous inward current that raises the membrane potential towards $V_{High}$ when the membrane is depolarized. Transistor $M_2$, analogous to the voltage-gated potassium channel in the Morris–Lecar model, provides a delayed outward current that lowers the membrane potential toward $V_{Low}$ when the membrane is depolarized. $V_H$ and $V_L$ are analogous to the half-activation voltages for the inward and outward currents, respectively. The voltages across $C_1$ and $C_2$ are the state variables representing the membrane potential, $V$, and the slow "activation" variable of the outward current, $W$, respectively. The $W$-nullcline represents its steady-state activation curve. Unlike the Morris–Lecar model, our silicon neuron model does not possess a leak current.

Using current conservation at node $V$, the net current charging $C_1$ is given by

$$C_1 \dot{V} = I_{ext}\alpha_P + i_H\alpha_P - i_L\alpha_N \qquad (1)$$

where $i_H$ and $i_L$ are the output currents of a differential pair circuit, and $\alpha_P$ and $\alpha_N$ describe the ohmic effects of transistors $M_1$ and $M_2$, respectively. The net current into $C_2$ is given by

$$C_2 \dot{W} = i_X\beta_P\beta_N \qquad (2)$$

where $i_X$ is the output current of the OTA, and $\beta_P$ and $\beta_N$ account for ohmic effects of the pull-up and the pull-down transistors inside the OTA.

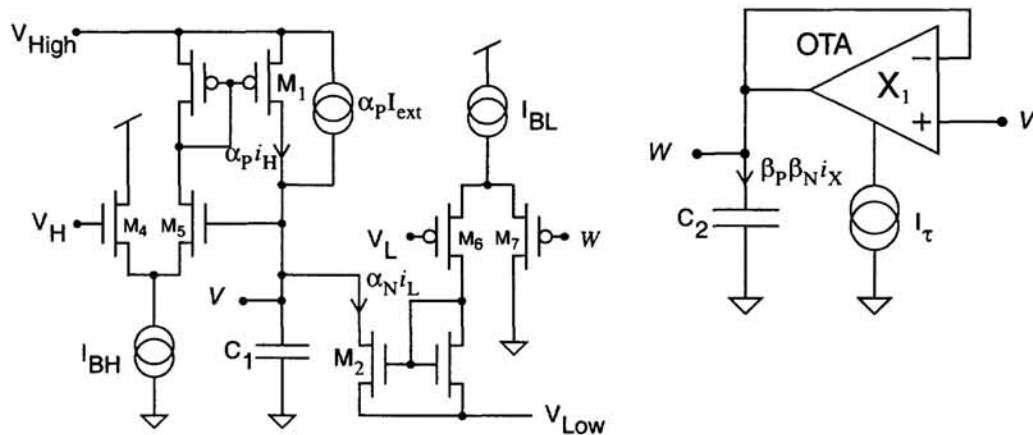

Figure 1: Circuit diagram of the silicon neuron. The circuit incorporates analog building blocks including two differential pair circuits composed of a bias current, $IB_H$, and transistors $M_4$–$M_5$, and a bias current, $IB_L$, and transistors $M_6$–$M_7$, and a single follower–integrator circuit composed of an operational transconductance amplifier (OTA), $X_1$ in the configuration shown and a load capacitor, $C_2$. The response of the follower–integrator circuit is similar to a first-order low-pass filter.

The output currents of the differential-pair and an OTA circuits, derived by using sub-threshold transistor equations [2], are a Fermi function and a hyperbolic-tangent function, respectively [2]. Substituting these functions for $i_H$, $i_L$, and $i_X$ in (1) and (2) yields

$$C_1 \dot{V} = I_{ext} \alpha_P + I_{BH} \frac{e^{\kappa(V - V_H)/U_T}}{1 + e^{\kappa(V - V_H)/U_T}} \alpha_P - I_{BL} \frac{e^{\kappa(W - V_L)/U_T}}{1 + e^{\kappa(W - V_L)/U_T}} \alpha_N$$

(3)

$$C_2 \dot{W} = I_\tau \tanh\left(\kappa \frac{V - W}{2 U_T}\right) \beta_P \beta_N$$

where

$$\alpha_P = 1 - e^{V - V_{High}/U_T} \qquad \alpha_N = 1 - e^{V_{Low} - V/U_T}$$

$$\beta_P = 1 - e^{W - V_{dd}/U_T} \qquad \beta_N = 1 - e^{-W/U_T}$$

(4)

$U_T$ is the thermal voltage, $V_{dd}$ is the supply voltage, and $\kappa$ is a fabrication dependent parameter. The terms $\alpha_P$ and $\alpha_N$ limit the range of $V$ to within $V_{High}$ and $V_{Low}$, and the terms $\beta_P$ and $\beta_N$ limit the range of $W$ to within the supply rails (Vdd and Gnd).

In order to compare the model to the experimental results, we needed to determine values for all of the model parameters. $V_{High}$, $V_{Low}$, $V_H$, $V_L$, and $V_{dd}$ were directly measured in experiments. The parameters $IB_H$ and $IB_L$ were measured by voltage-clamp experiments performed on the silicon neuron. At room temperature, $U_T \approx 0.025$ volts. The value of $\kappa \approx 0.65$ was estimated by measuring the slope of the steady-state activation curve of inward current. Because $W$ was implemented as an inaccessible node, $I_\tau$ could only be estimated. Based on the circuit design, we can assume that the bias currents $I_\tau$ and $I_{BH}$ are of the same order of magnitude. We choose $I_\tau \approx 2.2$ nA to fit the bifurcation diagram (see Figure 3). C1 and C2, which are assumed to be identical according to the physical design, are time scaling parameters in the model. We choose their values (C1 = C2 = 28 pF) to fit frequency dependence on $I_{ext}$ (see Figure 4).

## 3 Bifurcation analysis

The silicon neuron and the mathematical model[1] described by (3) demonstrate various dynamical behaviors under different parametric conditions. In particular, stable oscillations and steady-state equilibria are observed for different values of the externally applied current, $I_{ext}$. We focused our analysis on the influence of $I_{ext}$ on the neuron behavior for two reasons: (i) it provides insight about effects of synaptic currents, and (ii) it allows comparison with neurophysiological experiments in which polarizing current is used as a primary control parameter. The main results of this work are presented as the comparison between the mathematical models and the experimental data represented as bifurcation diagrams and frequency dependencies.

The nullclines described by (3) and for $I_{ext} = 32$ nA are shown in Figure 2A. In the regime that we operate the circuit, the $W$-nullcline is an almost-linear curve and the $V$-nullcline is an N-shaped curve. From (3), it can be seen that when $IB_H + I_{ext} > IB_L$ the nullclines cross at $(V, W) \approx (V_{High}, V_{High})$ and the system has high voltage (about 5 volts) steady-state equilibrium. Similarly, for $I_{ext}$ close to zero, the system has one stable equilibrium point close to $(V, W) \approx (V_{Low}, V_{Low})$.

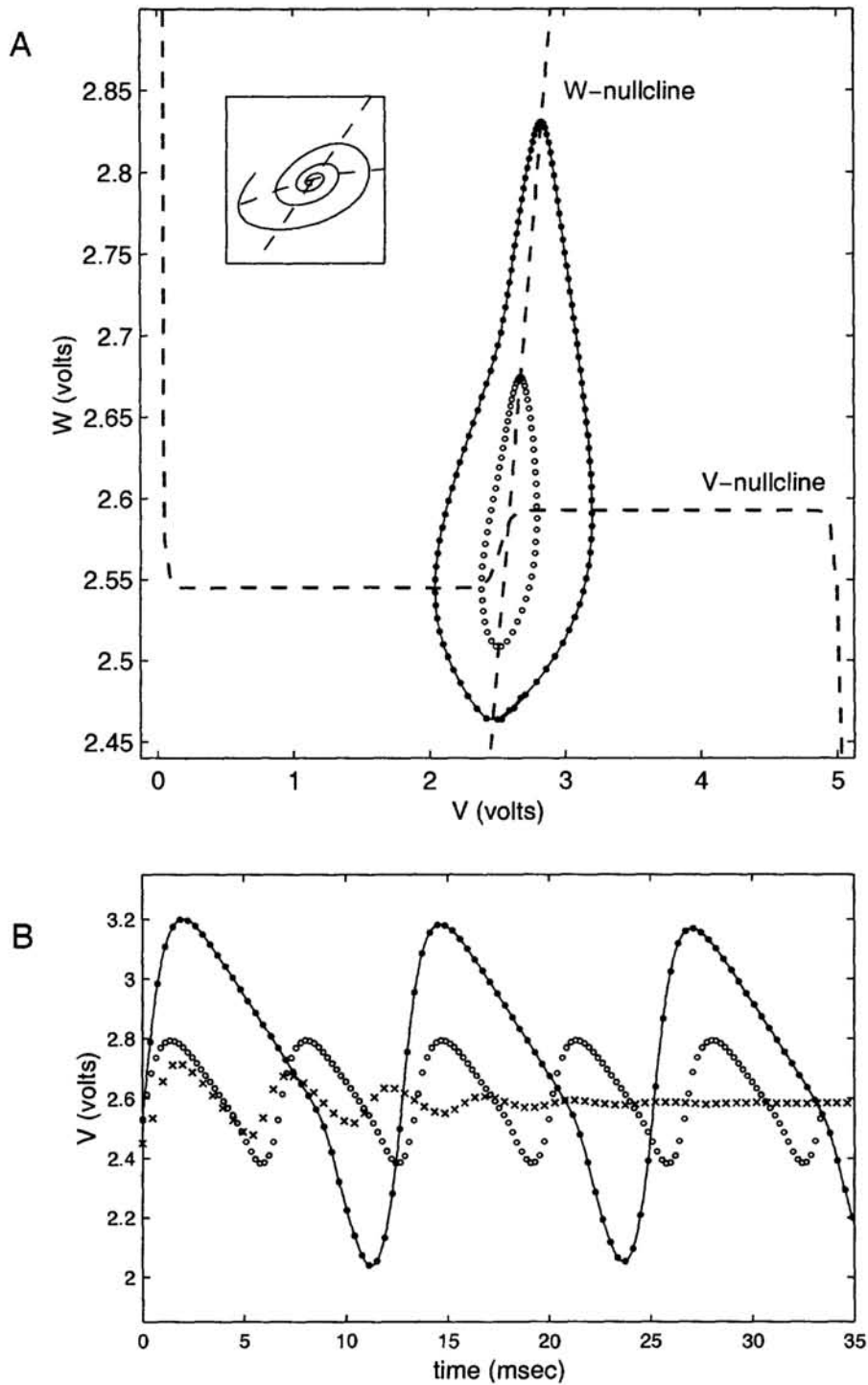

Figure 2: Nullclines and trajectories in the model of the silicon neuron for $I_{ext} = 32$ nA. The system exhibits a stable limit-cycle (filled circles), an unstable limit-cycle (unfilled circles), and stable equilibrium point. Unstable limit-cycle separates the basins of attraction of the stable limit-cycle and stable equilibrium point. Thus, trajectories initiated within the area bounded by the unstable limit-cycle approach the stable equilibrium point (solid line in A's inset, and "x's" in B). Trajectories initiated outside the unstable limit-cycle approach the stable limit-cycle. In A, the inset shows an expansion at the intersection of the $V$- and $W$-nullclines.

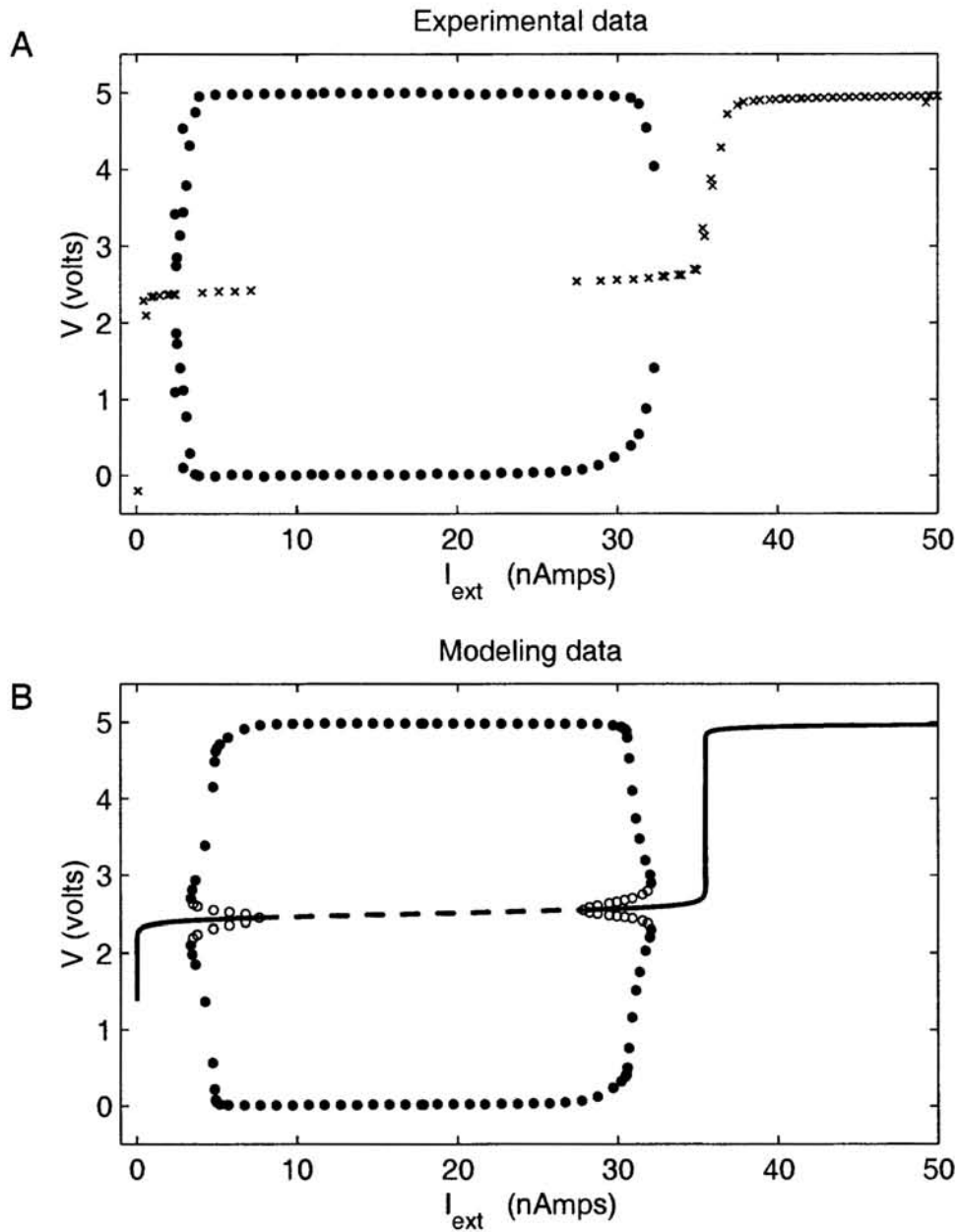

Figure 3: Bifurcation diagrams of the hardware implementation (A) and of the mathematical model (B) under variation of the externally applied current. In A, the steady-state equilibrium potential of $V$ is denoted by "x"s. The maximum and minimum values of $V$ during stable oscillations are denoted by the filled circles. In B, the stable and unstable equilibrium points are denoted by the solid and dashed curve, respectively, and the minimum and maximum values of the stable and unstable oscillations are denoted by the filled and unfilled circles, respectively. In B, limit-cycle oscillations appear and disappear via sub-critical Andronov–Hopf bifurcations. The bifurcation diagram (B) was computed with the LOCBIF program [14].

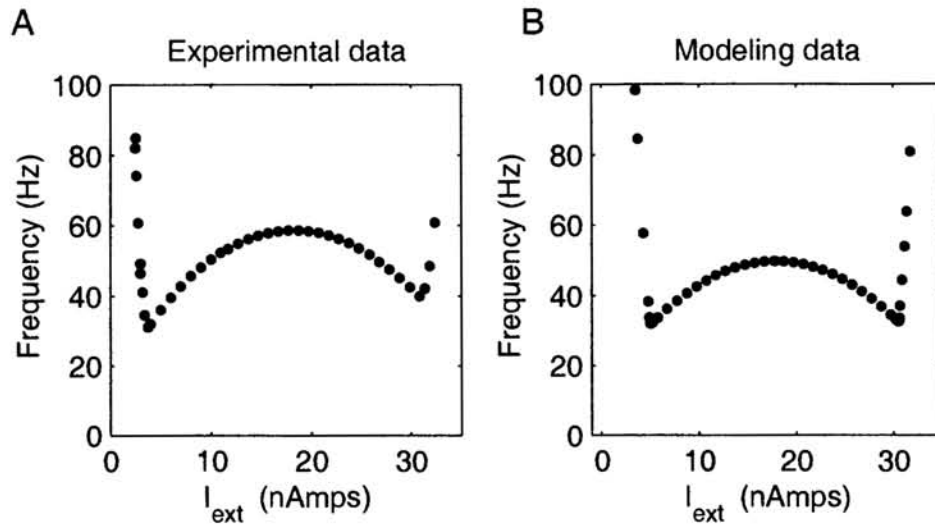

Figure 4: Frequency dependence of the silicon neuron (A) and the mathematical model (B) on the externally applied current.

For moderate values of $I_{ext}$ ([1 nA,34 nA]), the stable and unstable equilibrium points are close to $(V, W) \approx (V_H, V_L)$ (Figure 3). In experiments in which $I_{ext}$ was varied, we observed a hard loss of the stability of the steady-state equilibrium and a transition into oscillations at $I_{ext} = 7.2$ nA ($I_{ext} = 27.5$ nA). In the mathematical model, at the critical value of $I_{ext} = 7.7$ nA ($I_{ext} = 27.8$ nA), an unstable limit cycle appears via a subcritical Andronov-Hopf bifurcation. This unstable limit cycle merges with the stable limit cycle at the fold bifurcation at $I_{ext} = 3.4$ nA ($I_{ext} = 32.1$ nA). Similarly, in the experiments, we observed hard loss of stability of oscillations at $I_{ext} = 2.0$ nA ($I_{ext} = 32.8$ nA). Thus, the system demonstrates hysteresis. For example, when $I_{ext} = 20$ nA the silicon neuron has only one stable regime, namely, stable oscillations. Then if external current is slowly increased to $I_{ext} = 32.8$ nA, the form of oscillations changes. At this critical value of the current, the oscillations suddenly lose stability, and only steady-state equilibrium is stable. Now, when the external current is reduced, the steady-state equilibrium is observed at the values of the current where oscillations were previously exhibited. Thus, within the ranges of externally applied currents (2.0,7.2) and (27.5,32.8), oscillations and a steady-state equilibrium are stable regimes as shown in Figure 2.

## 4  Discussion

We have developed a two-state silicon neuron and a mathematical model that describes the behavior of this neuron. We have shown experimentally and verified mathematically that this silicon neuron has three regions of operation under the variation of its external current (one of its parameters). We also perform bifurcation analysis of the mathematical model by varying the externally applied current and show that the behaviors exhibited by the silicon neuron under corresponding conditions are in good agreement to those predicted by the bifurcation analysis.

This analysis and comparison to experiment is an important step toward our understanding of a variety of oscillatory hardware networks that we and others are developing. The

model facilitates an understanding of the neurons that the hardware alone does not provide. In particular for this neuron, the model allows us to determine the location of the unstable fixed points and the types of bifurcations that are exhibited. In higher-order systems, we expect that the model will provide us insight about observed behaviors and complex bifurcations in the phase space. The good matching between the model and the experimental data described in this paper gives us some confidence that future analysis efforts will prove fruitful.

## Acknowledgments

S. DeWeerth and G. Patel are funded by NSF grant IBN-9511721, G.S. Cymbalyuk is supported by Russian Foundation of Fundamental Research grant 99-04-49112, R.L. Calabrese and G.S. Cymbalyuk are supported by NIH grants NS24072 and NS34975.

## Footnotes

[1]The parameters used throughout the analyses of the model are $V_{Low} = 0$ V, $V_{High} = 5$ V, $V_L = V_H = 2.5$ V, $I_{BH} = 6.5$ nA, $I_{BL} = 42$ nA, $I_\tau = 2.2$ nA, $V_{dd} = 5$ V, $U_t = 0.025$ mV, and $\kappa = 0.65$.

## References

[1] Van Der Pol, B (1939) Biological rhythms considered as relaxation oscillations In H. Bremmer and C.J. Bouwkamp (eds) *Selected Scientific Papers*, Vol 2, North Holland Pub. Co., 1960.

[2] Mead, C.A. *Analog VLSI and Neural Systems*. Addison-Wesley, Reading, MA, 1989.

[3] Simoni, M.F., Patel, G.N., DeWeerth, S.P., & Calabrese, R.L. Analog VLSI model of the leech heartbeat elemental oscillator. Sixth Annual Computational Neuroscience Meeting, 1997. in Big Sky, Montana.

[4] DeWeerth, S., Patel, G., Schimmel, D., Simoni, M. and Calabrese, R. (1997). In *Proceedings of the Seventeenth Conference on Advanced Research in VLSI*, R.B. Brown and A.T. Ishii (eds), Los Alamitos, CA: IEEE Computer Society, 182–200.

[5] Marder, E. & Calabrese, R.L. (1996) Principles of rhythmic motor pattern generation. *Physiological Reviews* **76** (3): 687-717.

[6] Kopell, N. & Ermentrout, B. (1988) Coupled oscillators and the design of central pattern generators. *Mathematical biosciences* 90: 87-109.

[7] Skinner, F.K., Turrigiano, G.G., & Marder, E. (1993) Frequency and burst duration in oscillating neurons and two-cell networks. *Biological Cybernetics* **69**: 375-383.

[8] Skinner, F.K., Gramoll, S., Calabrese, R.L., Kopell, N. & Marder, E. (1994) Frequency control in biological half-center oscillators. In F.H. Eeckman (ed.), *Computation in neurons and neural systems*, pp. 223-228, Boston: Kluwer Academic Publishers.

[9] Patel, G. Holleman, J., DeWeerth, S. Analog VLSI model of intersegmental coordination with nearest-neighbor coupling. In , 1997.

[10] Patel, G. *A neuromorphic architecture for modelling intersegmental coordination*. Ph.D. dissertation, Georgia Institute of Technology, 1999.

[11] J. Guckenheimer and P. Holmes. *Nonlinear Oscillations, Dynamical Systems, and Bifurcation of Vector Fields*. Applied Mathematical Sciences, 42. Springer-Verlag, New York, New York, Heidelberg, Berlin, 1983.

[12] Morris, C. and Lecar, H. (1981) Voltage oscillations in the barnacle giant muscle fiber. *Biophys. J*, 35: 193–213.

[13] Rinzel, J. & Ermentrout, G.B. (1989) Analysis of Neural Excitability and Oscillations. In C. Koch and I. Segev (eds) *Methods in Neuronal Modeling from Synapses to Networks*. MIT press, Cambridge, MA.

[14] Khibnik, A.I., Kuznetsov, Yu.A., Levitin, V.V., Nikolaev, E.V. (1993) Continuation techniques and interactive software for bifurcation analysis of ODEs and iterated maps. *Physica D* **62** (1-4): 360-367.